# Identifying Dendritic Processing

**Aurel A. Lazar**
Department of Electrical Engineering
Columbia University
New York, NY 10027
aurel@ee.columbia.edu

**Yevgeniy B. Slutskiy**[*]
Department of Electrical Engineering
Columbia University
New York, NY 10027
ys2146@columbia.edu

## Abstract

In system identification both the input and the output of a system are available to an observer and an algorithm is sought to identify parameters of a hypothesized model of that system. Here we present a novel formal methodology for identifying dendritic processing in a neural circuit consisting of a linear dendritic processing filter in cascade with a spiking neuron model. The input to the circuit is an analog signal that belongs to the space of bandlimited functions. The output is a time sequence associated with the spike train. We derive an algorithm for identification of the dendritic processing filter and reconstruct its kernel with arbitrary precision.

## 1 Introduction

The nature of encoding and processing of sensory information in the visual, auditory and olfactory systems has been extensively investigated in the systems neuroscience literature. Many phenomenological [1, 2, 3] as well as mechanistic [4, 5, 6] models have been proposed to characterize and clarify the representation of sensory information on the level of single neurons.

Here we investigate a class of phenomenological neural circuit models in which the time-domain linear processing takes place in the dendritic tree and the resulting aggregate dendritic current is encoded in the spike domain by a spiking neuron. In block diagram form, these neural circuit models are of the [Filter]-[Spiking Neuron] type and as such represent a fundamental departure from the standard Linear-Nonlinear-Poisson (LNP) model that has been used to characterize neurons in many sensory systems, including vision [3, 7, 8], audition [2, 9] and olfaction [1, 10]. While the LNP model also includes a linear processing stage, it describes spike generation using an inhomogeneous Poisson process. In contrast, the [Filter]-[Spiking Neuron] model incorporates the temporal dynamics of spike generation and allows one to consider more biologically-plausible spike generators.

We perform identification of dendritic processing in the [Filter]-[Spiking Neuron] model assuming that input signals belong to the space of bandlimited functions, a class of functions that closely model natural stimuli in sensory systems. Under this assumption, we show that the identification of dendritic processing in the above neural circuit becomes mathematically tractable. Using simulated data, we demonstrate that under certain conditions it is possible to identify the impulse response of the dendritic processing filter with arbitrary precision. Furthermore, we show that the identification results fundamentally depend on the bandwidth of test stimuli.

The paper is organized as follows. The phenomenological neural circuit model and the identification problem are formally stated in section 2. The Neural Identification Machine and its realization as an algorithm for identifying dendritic processing is extensively discussed in section 3. Performance of the identification algorithm is exemplified in section 4. Finally, section 5 concludes our work.

---

[*]The names of the authors are alphabetically ordered.

## 2 Problem Statement

In what follows we assume that the dendritic processing is linear [11] and any nonlinear effects arise as a result of the spike generation mechanism [12]. We use linear BIBO-stable filters (not necessarily causal) to describe the computation performed by the dendritic tree. Furthermore, a spiking neuron model (as opposed to a rate model) is used to model the generation of action potentials or spikes.

We investigate a general neural circuit comprised of a filter in cascade with a spiking neuron model (Fig. 1(a)). This circuit is an instance of a Time Encoding Machine (TEM), a nonlinear asynchronous circuit that encodes analog signals in the time domain [13, 14]. Examples of spiking neuron models considered in this paper include the ideal IAF neuron, the leaky IAF neuron and the threshold-and-feedback (TAF) neuron [15]. However, the methodology developed below can be extended to many other spiking neuron models as well.

We break down the full identification of this circuit into two problems: (i) identification of linear operations in the dendritic tree and (ii) identification of spike generator parameters. First, we consider problem (i) and assume that parameters of the spike generator can be obtained through biophysical experiments. Then we show how to address (ii) by exploring the space of input signals. We consider a specific example of a neural circuit in Fig. 1(a) and carry out a full identification of that circuit.

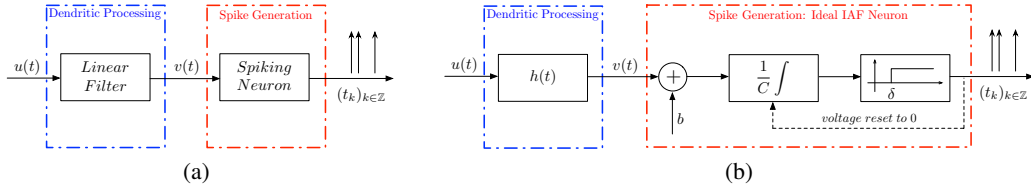

(a)                                                                 (b)

Figure 1: **Problem setup. (a)** The dendritic processing is described by a linear filter and spikes are produced by a (nonlinear) spiking neuron model. **(b)** An example of a neural circuit in (a) is a linear filter in cascade with the ideal IAF neuron. An input signal $u$ is first passed through a filter with an impulse response $h$. The output of the filter $v(t) = (u * h)(t), t \in \mathbb{R}$, is then encoded into a time sequence $(t_k)_{k \in \mathbb{Z}}$ by the ideal IAF neuron.

## 3 Neuron Identification Machines

A Neuron Identification Machine (NIM) is the realization of an algorithm for the identification of the dendritic processing filter in cascade with a spiking neuron model. First, we introduce several definitions needed to formally address the problem of identifying dendritic processing. We then consider the [Filter]-[Ideal IAF] neural circuit. We derive an algorithm for a perfect identification of the impulse response of the filter and provide conditions for the identification with arbitrary precision. Finally, we extend our results to the [Filter]-[Leaky IAF] and [Filter]-[TAF] neural circuits.

### 3.1 Preliminaries

We model signals $u = u(t), t \in \mathbb{R}$, at the input to a neural circuit as elements of the Paley-Wiener space $\Xi = \{u \in \mathbb{L}^2(\mathbb{R}) \mid \text{supp}(\mathcal{F}u) \subseteq [-\Omega, \Omega]\}$, i.e., as functions of finite energy having a finite spectral support ($\mathcal{F}$ denotes the Fourier transform). Furthermore, we assume that the dendritic processing filters $h = h(t), t \in \mathbb{R}$, are linear, BIBO-stable and have a finite temporal support, i.e., they belong to the space $H = \{h \in \mathbb{L}^1(\mathbb{R}) \mid \text{supp}(h) \subseteq [T_1, T_2]\}$.

**Definition 1.** *A signal $u \in \Xi$ at the input to a neural circuit together with the resulting output $\mathbb{T} = (t_k)_{k \in \mathbb{Z}}$ of that circuit is called an input/output (I/O) pair and is denoted by $(u, \mathbb{T})$.*

**Definition 2.** *Two neural circuits are said to be $\Xi$-I/O-equivalent if their respective I/O pairs are identical for all $u \in \Xi$.*

**Definition 3.** *Let $\mathcal{P} : H \to \Xi$ with $(\mathcal{P}h)(t) = (h * g)(t)$, where $(h * g)$ denotes the convolution of $h$ with the sinc kernel $g \triangleq \sin(\Omega t)/(\pi t), t \in \mathbb{R}$. We say that $\mathcal{P}h$ is the projection of $h$ onto $\Xi$.*

**Definition 4.** *Signals $\{u^i\}_{i=1}^N$ are said to be linearly independent if there do not exist real numbers $\{\alpha_i\}_{i=1}^N$, not all zero, and real numbers $\{\beta_i\}_{i=1}^N$ such that $\sum_{i=1}^N \alpha_i u^i(t + \beta_i) = 0$.*

## 3.2 NIM for the [Filter]-[Ideal IAF] Neural Circuit

An example of a model circuit in Fig. 1(a) is the [Filter]-[Ideal IAF] circuit shown in Fig. 1(b). In this circuit, an input signal $u \in \Xi$ is passed through a filter with an impulse response (kernel) $h \in H$ and then encoded by an ideal IAF neuron with a bias $b \in \mathbb{R}_+$, a capacitance $C \in \mathbb{R}_+$ and a threshold $\delta \in \mathbb{R}_+$. The output of the circuit is a sequence of spike times $(t_k)_{k \in \mathbb{Z}}$ that is available to an observer. This neural circuit is an instance of a TEM and its operation can be described by a set of equations (formally known as the t-transform [13]):

$$\int_{t_k}^{t_{k+1}} (u * h)(s)ds = q_k, \quad k \in \mathbb{Z}, \tag{1}$$

where $q_k \triangleq C\delta - b(t_{k+1} - t_k)$. Intuitively, at every spike time $t_{k+1}$ the ideal IAF neuron is providing a measurement $q_k$ of the signal $v(t) = (u * h)(t)$ on the interval $t \in [t_k, t_{k+1}]$.

**Proposition 1.** *The left-hand side of the t-transform in (1) can be written as a bounded linear functional* $L_k : \Xi \to \mathbb{R}$ *with* $L_k(\mathcal{P}h) = \langle \phi_k, \mathcal{P}h \rangle$, *where* $\phi_k(t) = \left( 1_{[t_k, t_{k+1}]} * \tilde{u} \right)(t)$ *and* $\tilde{u} = u(-t)$, $t \in \mathbb{R}$, *denotes the involution of* $u$.

**Proof:** Since $(u * h) \in \Xi$, we have $(u * h)(t) = (u * h * g)(t)$, $t \in \mathbb{R}$, and therefore $\int_{t_k}^{t_{k+1}} (u * h)(s)ds = \int_{t_k}^{t_{k+1}} (u * \mathcal{P}h)(s)ds$. Now since $\mathcal{P}h$ is bounded, the expression on the right-hand side of the equality is a bounded linear functional $L_k : \Xi \to \mathbb{R}$ with

$$L_k(\mathcal{P}h) = \int_{t_k}^{t_{k+1}} (u * \mathcal{P}h)(s)ds = \langle \phi_k, \mathcal{P}h \rangle, \tag{2}$$

where $\phi_k \in \Xi$ and the last equality follows from the Riesz representation theorem [16]. To find $\phi_k$, we use the fact that $\Xi$ is a Reproducing Kernel Hilbert Space (RKHS) [17] with a kernel $K(s,t) = g(t-s)$. By the reproducing property of the kernel [17], we have $\phi_k(t) = \langle \phi_k, K_t \rangle = L_k(K_t)$. Letting $\tilde{u} = u(-t)$ denote the involution of $u$ and using (2), we obtain

$$\phi_k(t) = \langle 1_{[t_k, t_{k+1}]} * \tilde{u}, \ K_t \rangle = \left( 1_{[t_k, t_{k+1}]} * \tilde{u} \right)(t).$$

$\square$

Proposition 1 effectively states that the measurements $(q_k)_{k \in \mathbb{Z}}$ of $v(t) = (u * h)(t)$ can be also interpreted as the measurements of $(\mathcal{P}h)(t)$. A natural question then is how to identify $\mathcal{P}h$ from $(q_k)_{k \in \mathbb{Z}}$. To that end, we note that an observer can typically record both the input $u = u(t)$, $t \in \mathbb{R}$ and the output $\mathbb{T} = (t_k)_{k \in \mathbb{Z}}$ of a neural circuit. Since $(q_k)_{k \in \mathbb{Z}}$ can be evaluated from $(t_k)_{k \in \mathbb{Z}}$ using the definition of $q_k$ in (1), the problem is reduced to identifying $\mathcal{P}h$ from an I/O pair $(u, \mathbb{T})$.

**Theorem 1.** *Let* $u$ *be bounded with* $\mathrm{supp}(\mathcal{F}u) = [-\Omega, \Omega]$, $h \in H$ *and* $b/(C\delta) > \Omega/\pi$. *Then given an I/O pair* $(u, \mathbb{T})$ *of the [Filter]-[Ideal IAF] neural circuit,* $\mathcal{P}h$ *can be perfectly identified as*

$$(\mathcal{P}h)(t) = \sum_{k \in \mathbb{Z}} c_k \psi_k(t),$$

*where* $\psi_k(t) = g(t - t_k)$, $t \in \mathbb{R}$. *Furthermore,* $\mathbf{c} = \mathbf{G}^+ \mathbf{q}$ *with* $\mathbf{G}^+$ *denoting the Moore-Penrose pseudoinverse of* $\mathbf{G}$, $[\mathbf{G}]_{lk} = \int_{t_l}^{t_{l+1}} u(s - t_k)ds$ *for all* $k, l \in \mathbb{Z}$, *and* $[\mathbf{q}]_l = C\delta - b(t_{l+1} - t_l)$.

**Proof:** By appropriately bounding the input signal $u$, the spike density (the average number of spikes over arbitrarily long time intervals) of an ideal IAF neuron is given by $D = b/(C\delta)$ [14]. Therefore, for $D > \Omega/\pi$ the set of the representation functions $(\psi_k)_{k \in \mathbb{Z}}$, $\psi_k(t) = g(t - t_k)$, is a frame in $\Xi$ [18] and $(\mathcal{P}h)(t) = \sum_{k \in \mathbb{Z}} c_k \psi_k(t)$. To find the coefficients $c_k$ we note from (2) that

$$q_l = \langle \phi_l, \mathcal{P}h \rangle = \sum_{k \in \mathbb{Z}} c_k \langle \phi_l, \psi_k \rangle = \sum_{k \in \mathbb{Z}} [\mathbf{G}]_{lk} c_k, \tag{3}$$

where $[\mathbf{G}]_{lk} = \langle \phi_l, \psi_k \rangle = \langle 1_{[t_l, t_{l+1}]} * \tilde{u}, \ g(\cdot - t_k) \rangle = \int_{t_l}^{t_{l+1}} u(s - t_k)ds$. Writing (3) in matrix form, we obtain $\mathbf{q} = \mathbf{Gc}$ with $[\mathbf{q}]_l = q_l$ and $[\mathbf{c}]_k = c_k$. Finally, the coefficients $c_k$, $k \in \mathbb{Z}$, can be computed as $\mathbf{c} = \mathbf{G}^+ \mathbf{q}$. $\square$

**Remark 1.** *The condition $b/(C\delta) > \Omega/\pi$ in Theorem 1 is a Nyquist-type rate condition. Thus, perfect identification of the projection of $h$ onto $\Xi$ can be achieved for a finite average spike rate.*

**Remark 2.** *Ideally, we would like to identify the kernel $h \in H$ of the filter in cascade with the ideal IAF neuron. Note that unlike $h$, the projection $\mathcal{P}h$ belongs to the space $\mathbb{L}^2(\mathbb{R})$, i.e., in general $\mathcal{P}h$ is not BIBO-stable and does not have a finite temporal support. Nevertheless, it is easy to show that $(\mathcal{P}h)(t)$ approximates $h(t)$ arbitrarily closely on $t \in [T_1, T_2]$, provided that the bandwidth $\Omega$ of $u$ is sufficiently large.*

**Remark 3.** *If the impulse response $h(t) = \delta(t)$, i.e., if there is no processing on the (arbitrary) input signal $u(t)$, then $q_l = \int_{t_l}^{t_{l+1}} (u * h)(s) ds = \int_{t_l}^{t_{l+1}} u(s) ds, l \in \mathbb{Z}$. Furthermore,*

$$\int_{t_l}^{t_{l+1}} (u * \mathcal{P}h)(s) ds = \int_{t_l}^{t_{l+1}} (u * h)(s) ds = \int_{t_l}^{t_{l+1}} u(s) ds = \int_{t_l}^{t_{l+1}} (u * g)(s) ds, \qquad l \in \mathbb{Z}.$$

*The above holds if and only if $(\mathcal{P}h)(t) = g(t)$, $t \in \mathbb{R}$. In other words, if $h(t) = \delta(t)$, then we identify $\mathcal{P}\delta(t) = \sin(\Omega t)/(\pi t)$, the projection of $\delta(t)$ onto $\Xi$.*

**Corollary 1.** *Let $u$ be bounded with $\mathrm{supp}(\mathcal{F}u) = [-\Omega, \Omega]$, $h \in H$ and $\frac{b}{C\delta} > \frac{\Omega}{\pi}$. Furthermore, let $W = (\tau_1, \tau_2)$ so that $(\tau_2 - \tau_1) > (T_2 - T_1)$ and let $\tau = (\tau_1 + \tau_2)/2$, $T = (T_1 + T_2)/2$. Then given an I/O pair $(u, \mathbb{T})$ of the [Filter]-[Ideal IAF] neural circuit, $(\mathcal{P}h)(t)$ can be approximated arbitrarily closely on $t \in [T_1, T_2]$ by*

$$\hat{h}(t) = \sum_{k: t_k \in W} c_k \psi_k(t),$$

*where $\psi_k(t) = g(t - (t_k - \tau + T))$, $\mathbf{c} = \mathbf{G}^+\mathbf{q}$, $[\mathbf{G}]_{lk} = \int_{t_l}^{t_{l+1}} u(s - (t_k - \tau + T)) ds$ and $[\mathbf{q}]_l = C\delta - b(t_{l+1} - t_l)$ for all $k, l \in \mathbb{Z}$, provided that $|\tau_1|$ and $|\tau_2|$ are sufficiently large.*

**Proof:** Through a change of coordinates $t \to t' = (t - \tau + T)$ illustrated in Fig. 2, we obtain $W' = [\tau_1 - \tau + T, \ \tau_2 - \tau + T] \supset [T_1, T_2]$ and the set of spike times $(t_k - \tau + T)_{k: t_k \in W}$. Note that $W' \to \mathbb{R}$ as $(\tau_2 - \tau_1) \to \infty$. The rest of the proof follows from Theorem 1 and the fact that $\lim_{t \to \pm\infty} g(t) = 0$. $\qquad\square$

From Corollary 1 we see that if the [Filter]-[Ideal IAF] neural circuit is producing spikes with a spike density above the Nyquist rate, then we can use a set of spike times $(t_k)_{k: t_k \in W}$ from a single temporal window $W$ to identify $(\mathcal{P}h)(t)$ to an arbitrary precision on $[T_1, T_2]$.

This result is not surprising. Since the spike density is above the Nyquist rate, we could have also used a canonical time decoding machine (TDM) [13] to first perfectly recover the filter output $v(t)$ and then employ one of the widely available LTI system techniques to estimate $(\mathcal{P}h)(t)$.

However, the problem becomes much more difficult if the spike density is below the Nyquist rate.

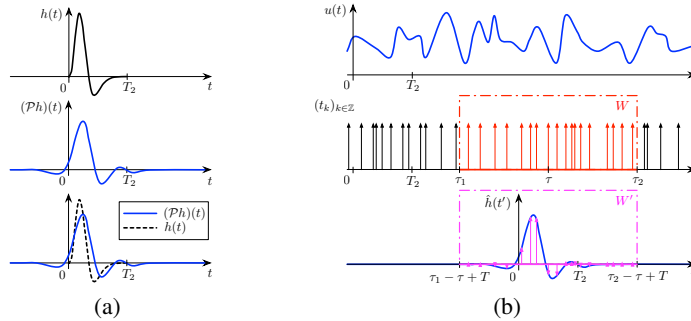

(a)                                              (b)

Figure 2: **Change of coordinates in Corollary 1. (a)** Top: example of a causal impulse response $h(t)$ with $\mathrm{supp}(h) = [T_1, T_2]$, $T_1 = 0$. Middle: projection $\mathcal{P}h$ of $h$ onto some $\Xi$. Note that $\mathcal{P}h$ is not causal and $\mathrm{supp}(\mathcal{P}h) = \mathbb{R}$. Bottom: $h(t)$ and $(\mathcal{P}h)(t)$ are plotted on the same set of axes. **(b)** Top: an input signal $u(t)$ with $\mathrm{supp}(\mathcal{F}u) = [-\Omega, \Omega]$. Middle: only red spikes from a temporal window $W = (\tau_1, \tau_2)$ are used to construct $\hat{h}(t)$. Bottom: $\mathcal{P}h$ is approximated by $\hat{h}(t)$ on $t \in [T_1, T_2]$ using spike times $(t_k - \tau + T)_{k: t_k \in W}$.

**Theorem 2.** *(The Neuron Identification Machine) Let $\{u^i \mid \text{supp}(\mathcal{F}u^i) = [-\Omega, \Omega]\}_{i=1}^N$ be a collection of $N$ linearly independent and bounded stimuli at the input to a [Filter]-[Ideal IAF] neural circuit with a dendritic processing filter $h \in H$. Furthermore, let $\mathbb{T}^i = (t_k^i)_{k \in \mathbb{Z}}$ denote the output of the neural circuit in response to the bounded input signal $u^i$. If $\sum_{j=1}^N \frac{b}{C\delta} > \frac{\Omega}{\pi}$, then $(\mathcal{P}h)(t)$ can be identified perfectly from the collection of I/O pairs $\{(u^i, \mathbb{T}^i)\}_{i=1}^N$.*

**Proof:** Consider the SIMO TEM [14] depicted in Fig. 3(a). $h(t)$ is the input to a population of $N$ [Filter]-[Ideal IAF] neural circuits. The spikes $(t_k^i)_{k \in \mathbb{Z}}$ at the output of each neural circuit represent distinct measurements $q_k^i = \langle \phi_k^i, \mathcal{P}h \rangle$ of $(\mathcal{P}h)(t)$. Thus we can think of the $q_k^i$'s as projections of $\mathcal{P}h$ onto $(\phi_1^1, \phi_2^1, \ldots, \phi_k^1, \ldots, \phi_1^N, \phi_2^N, \ldots, \phi_k^N, \ldots)$. Since the filters are linearly independent [14], it follows that, if $\{u^i\}_{i=1}^N$ are appropriately bounded and $\sum_{j=1}^N \frac{b}{C\delta} > \frac{\Omega}{\pi}$ or equivalently if the number of neurons $N > \frac{\Omega C \delta}{\pi b} = \frac{\Omega}{\pi D}$, the set of functions $\{(\psi_k^j)_{k \in \mathbb{Z}}\}_{j=1}^N$ with $\psi_k^j(t) = g(t - t_k^j)$, is a frame for $\Xi$ [14], [18]. Hence

$$(\mathcal{P}h)(t) = \sum_{j=1}^N \sum_{k \in \mathbb{Z}} c_k^j \psi_k^j(t). \tag{4}$$

To find the coefficients $c_k$, we take the inner product of (4) with $\phi_l^1(t), \phi_l^2(t), \ldots, \phi_l^N(t)$:

$$\langle \phi_l^i, \mathcal{P}h \rangle = \sum_{k \in \mathbb{Z}} c_k^1 \langle \phi_l^i, \psi_k^1 \rangle + \sum_{k \in \mathbb{Z}} c_k^2 \langle \phi_l^i, \psi_k^2 \rangle + \cdots + \sum_{k \in \mathbb{Z}} c_k^N \langle \phi_l^i, \psi_k^N \rangle \equiv q_l^i,$$

for $i = 1, \ldots, N, l \in \mathbb{Z}$. Letting $[\mathbf{G}^{ij}]_{lk} = \langle \phi_l^i, \psi_k^j \rangle$, we obtain

$$q_l^i = \sum_{k \in \mathbb{Z}} [\mathbf{G}^{i1}]_{lk} c_k^1 + \sum_{k \in \mathbb{Z}} [\mathbf{G}^{i2}]_{lk} c_k^2 + \cdots + \sum_{k \in \mathbb{Z}} [\mathbf{G}^{iN}]_{lk} c_k^N, \tag{5}$$

for $i = 1, \ldots, N, l \in \mathbb{Z}$. Writing (5) in matrix form, we have $\mathbf{q} = \mathbf{Gc}$, where $\mathbf{q} = [\mathbf{q}^1, \mathbf{q}^2, \ldots, \mathbf{q}^N]^T$ with $[\mathbf{q}^i]_l = C\delta - b(t_{l+1}^i - t_l^i)$, $[\mathbf{G}^{ij}]_{lk} = \int_{t_l^i}^{t_{l+1}^i} u^i(s - t_k^j)ds$ and $\mathbf{c} = [\mathbf{c}^1, \mathbf{c}^2, \ldots, \mathbf{c}^N]^T$. Finally, to find the coefficients $c_k, k \in \mathbb{Z}$, we compute $\mathbf{c} = \mathbf{G}^+ \mathbf{q}$. $\qquad \square$

**Corollary 2.** *Let $\{u^i\}_{i=1}^N$ as before, $h \in H$ and $\sum_{j=1}^N \frac{b}{C\delta} > \frac{\Omega}{\pi}$. Furthermore, let $W = (\tau_1, \tau_2)$ so that $(\tau_2 - \tau_1) > (T_2 - T_1)$ and let $\tau = (\tau_1 + \tau_2)/2$, $T = (T_1 + T_2)/2$. Then given the I/O pairs $\{(u^i, \mathbb{T}^i)\}_{i=1}^N$ of the [Filter]-[Ideal IAF] neural circuit, $(\mathcal{P}h)(t)$ can be approximated arbitrarily closely on $t \in [T_1, T_2]$ by $\hat{h}(t) = \sum_{j=1}^N \sum_{k: t_k^j \in W} c_k^j \psi_k^j(t)$, where $\psi_k^j(t) = g(t - (t_k^j - \tau + T))$, $\mathbf{c} = \mathbf{G}^+ \mathbf{q}$, with $[\mathbf{G}^{ij}]_{lk} = \int_{t_l^i}^{t_{l+1}^i} u^i(s - (t_k^j - \tau + T))ds$, $\mathbf{q} = [\mathbf{q}^1, \mathbf{q}^2, \ldots, \mathbf{q}^N]^T$, $[\mathbf{q}^i]_l = C\delta - b(t_{l+1}^i - t_l^i)$ for all $k, l \in \mathbb{Z}$ provided that $|\tau_1|$ and $|\tau_2|$ are sufficiently large.*

**Proof:** Similar to Corollary 1. $\qquad \square$

**Corollary 3.** *Let $\text{supp}(\mathcal{F}u) = [-\Omega, \Omega]$, $h \in H$ and let $\{W^i \triangleq (\tau_1^i, \tau_2^i)\}_{i=1}^N$ be a collection of windows of fixed length $(\tau_2^i - \tau_1^i) > (T_2 - T_1), i = 1, 2, \ldots, N$. Furthermore, let $\tau^i = (\tau_1^i + \tau_2^i)/2$, $T = (T_1 + T_2)/2$ and let $(t_k^i)_{k \in \mathbb{Z}}$ denote those spikes of the I/O pair $(u, \mathbb{T})$ that belong to $W^i$. Then $\mathcal{P}h$ can be approximated arbitrarily closely on $[T_1, T_2]$ by*

$$\hat{h}(t) = \sum_{j=1}^N \sum_{k: t_k \in W^j} c_k^j \psi_k^j(t),$$

*where $\psi_k^j(t) = g(t - (t_k^j - \tau^j + T))$, $\mathbf{c} = \mathbf{G}^+ \mathbf{q}$ with $[\mathbf{G}^{ij}]_{lk} = \int_{t_l^i}^{t_{l+1}^i} u(s - (t_k^j - \tau^j + T))ds$, $\mathbf{q} = [\mathbf{q}^1, \mathbf{q}^2, \ldots, \mathbf{q}^N]^T$, $[\mathbf{q}^i]_l = C\delta - b(t_{l+1}^i - t_l^i)$ for all $k, l \in \mathbb{Z}$, provided that the number of non-overlapping windows $N$ is sufficiently large.*

**Proof:** The input signal $u$ restricted, respectively, to the collection of intervals $\{W^i \triangleq (\tau_1^i, \tau_2^i)\}_{i=1}^N$ plays the same role here as the test stimuli $\{u^i\}_{i=1}^N$ in Corollary 2. See also Remark 9 in [14]. $\qquad \square$

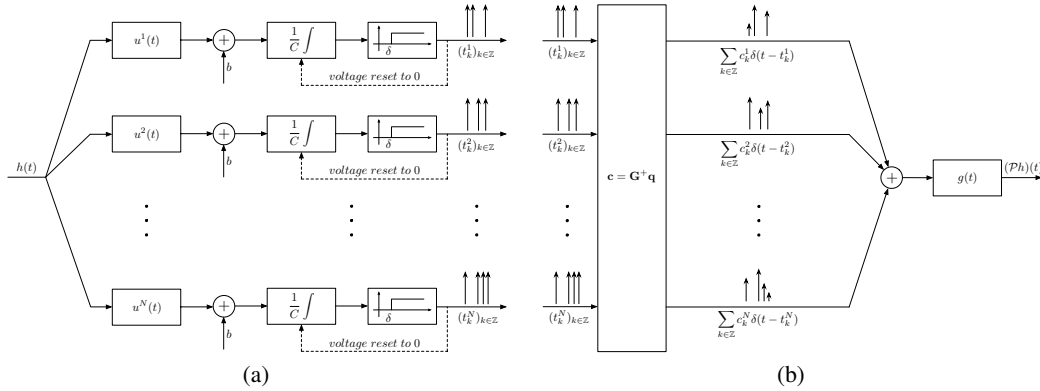

Figure 3: **The Neuron Identification Machine.** **(a)** SIMO TEM interpretation of the identification problem with $(t_k^i) = (t_k)_{k:t_k \in W^i}$, $i = 1, 2, \ldots, N$. **(b)** Block diagram of the algorithm in Theorem 2.

**Remark 4.** *The methodology presented in Theorem 2 can easily be applied to other spiking neuron models. For example, for the leaky IAF neuron, we have*

$$[\mathbf{q}^i]_l = C\delta - bRC\left[1 - \exp\left(\frac{t_l^i - t_{l+1}^i}{RC}\right)\right], \quad [\mathbf{G}^{ij}]_{lk} = \int_{t_l^i}^{t_{l+1}^i} u^i\left(s - t_k^j\right)\exp\left(\frac{s - t_{l+1}^i}{RC}\right)ds.$$

*Similarly, for a threshold-and-feedback (TAF) neuron [15] with a bias $b \in \mathbb{R}_+$, a threshold $\delta \in \mathbb{R}_+$, and a causal feedback filter with an impulse response $f(t)$, $t \in \mathbb{R}$, we obtain*

$$[\mathbf{q}^i]_l = \delta - b + \sum_{k<l} f(t_l^i - t_k^i), \qquad [\mathbf{G}^{ij}]_{lk} = u^i\left(t_l^i - t_k^j\right).$$

### 3.3   Identifying Parameters of the Spiking Neuron Model

If parameters of the spiking neuron model cannot be obtained through biophysical experiments, we can use additional input stimuli to derive a neural circuit that is $\Xi$-I/O-equivalent to the original circuit. For example, consider the circuit in Fig. 1(a). Rewriting the t-transform in (1), we obtain

$$\frac{1}{b}\int_{t_k}^{t_{k+1}} (u * h)(s)ds = \frac{C\delta}{b} - (t_{k+1} - t_k) \qquad \Longleftrightarrow \qquad \int_{t_k}^{t_{k+1}} (u * h')(s)ds = q_k',$$

where $h'(t) = h(t)/b$, $t \in \mathbb{R}$ and $q_k' = C\delta/b - (t_{k+1} - t_k)$.

Setting $u = 0$, we can now compute $C\delta/b = (t_{k+1} - t_k)$. Next we can use the NIM described in Section 3.2 to identify with arbitrary precision the projection $\mathcal{P}h'$ of $h'$ onto $\Xi$. Thus we identify a [Filter]-[Ideal IAF] circuit with a filter impulse response $\mathcal{P}h'$, a bias $b' = 1$, a capacitance $C' = 1$ and a threshold $\delta' = C\delta/b$. This neural circuit is $\Xi$-I/O-equivalent to the circuit in Fig. 1(b).

## 4   Examples

We now demonstrate the performance of the identification algorithm in Corollary 3. We model the dendritic processing filter using a causal linear kernel $h(t) = ce^{-\alpha t}\left[(\alpha t)^3/3! - (\alpha t)^5/5!\right]$ with $t \in [0, 0.1\,\text{s}]$, $c = 3$ and $\alpha = 200$. The general form of this kernel was suggested in [19] as a plausible approximation to the temporal structure of a visual receptive field.

We use two different bandlimited signals and show that the identification results fundamentally depend on the signal bandwidth $\Omega$. In Fig. 4 the signal is bandlimited to $\Omega = 2\pi \cdot 25\,\text{rad/s}$, whereas in Fig. 5 it is bandlimited to $\Omega = 2\pi \cdot 100\,\text{rad/s}$. Although in principle the kernel $h$ has an infinite bandwidth (having a finite temporal support), its effective bandwidth $\Omega \approx 2\pi \cdot 100\,\text{rad/s}$ (Fig. 6(b)). Thus in Fig. 4 we reconstruct the projection $\mathcal{P}h$ of the kernel $h$ onto $\Xi$ with $\Omega = 2\pi \cdot 25\,\text{rad/s}$, whereas in Fig. 5 we reconstruct nearly $h$ itself.

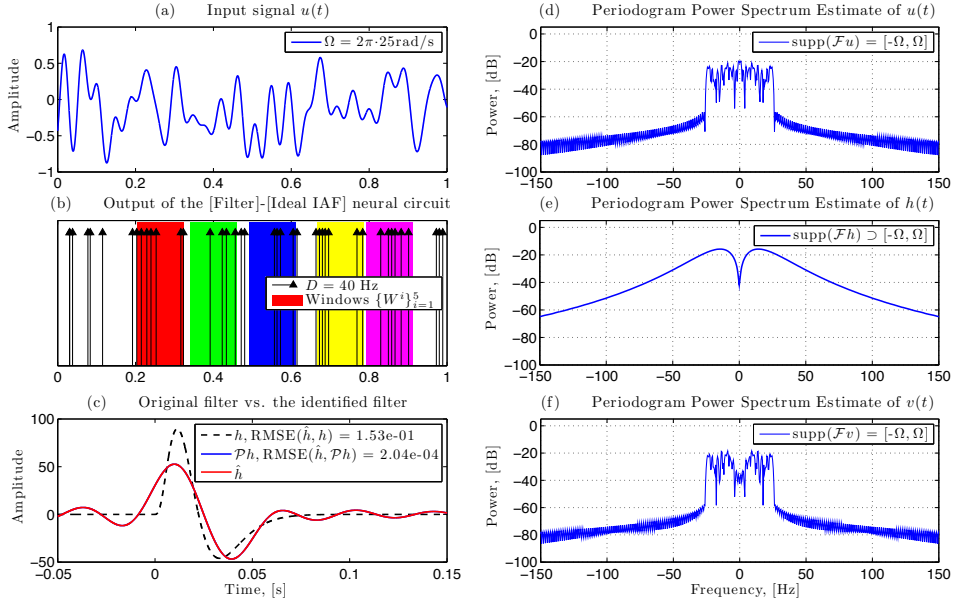

Figure 4: **Identifying dendritic processing in the [Filter]-[Ideal IAF] neural circuit.** $\Omega = 2\pi \cdot 25$ **rad/s.**
(a) Signal $u(t)$ at the input to the circuit. (b) The output of the circuit is a set of spikes at times $(t_k)_{k \in \mathbb{Z}}$. The
spike density $D = 40$ Hz. Note that only 25 spikes from 5 temporal windows are used to construct $\hat{h}$. (c) The
RMSE between $\hat{h}$ (red) and $\mathcal{P}h$ (blue) is $2.04 \times 10^{-4}$. The RMSE between $\hat{h}$ (red) and $h$ (dashed black) is
$1.53 \times 10^{-1}$. (d)-(f) Spectral estimates of $u$, $h$ and $v = u * h$. Note that $\text{supp}(\mathcal{F}u) = [-\Omega, \Omega] = \text{supp}(\mathcal{F}v)$
but $\text{supp}(\mathcal{F}h) \supset [-\Omega, \Omega]$. In other words, both $u, v \in \Xi$ but $h \notin \Xi$.

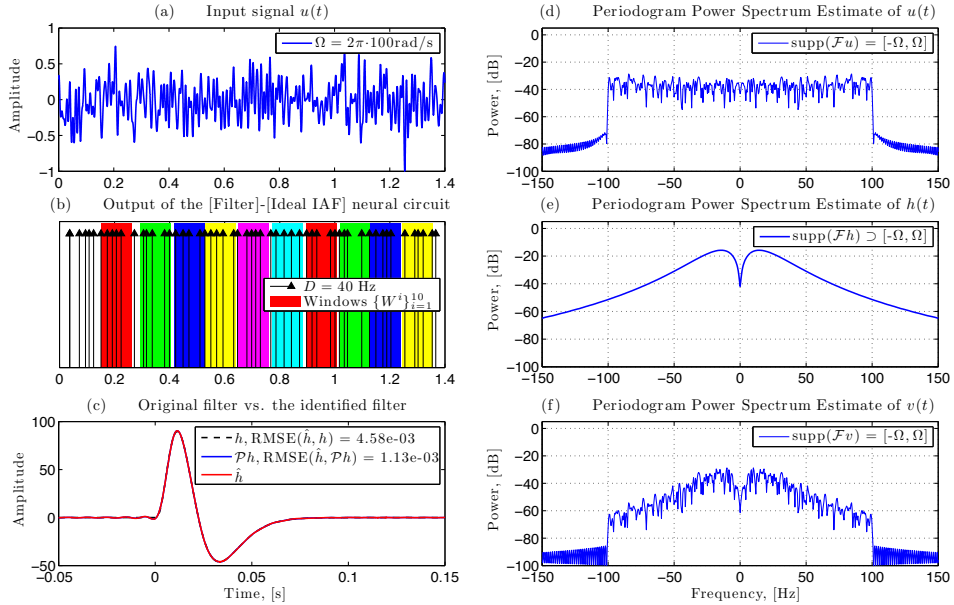

Figure 5: **Identifying dendritic processing of the [Filter]-[Ideal IAF] neural circuit.** $\Omega = 2\pi \cdot 100$ **rad/s.**
(a) Signal $u(t)$ at the input to the circuit. (b) The output of the circuit is a set of spikes at times $(t_k)_{k \in \mathbb{Z}}$. The
spike density $D = 40$ Hz. Note that only 43 spikes from 10 temporal windows are used to construct $\hat{h}$. (c) The
RMSE between $\hat{h}$ (red) and $\mathcal{P}h$ (blue) is $1.13 \times 10^{-3}$. The RMSE between $\hat{h}$ (red) and $h$ (dashed black) is
$4.58 \times 10^{-3}$. (d)-(f) Spectral estimates of $u$, $h$ and $v = u * h$. Note that $\text{supp}(\mathcal{F}u) = [-\Omega, \Omega] = \text{supp}(\mathcal{F}v)$
but $\text{supp}(\mathcal{F}h) \supset [-\Omega, \Omega]$. In other words, both $u, v \in \Xi$ but $h \notin \Xi$.

Next, we evaluate the filter identification error as a function of the number of temporal windows $N$ and the stimulus bandwidth $\Omega$. By increasing $N$, we can approximate the projection $\mathcal{P}h$ of $h$ with arbitrary precision (Fig. 6(a)). Note that the estimate $\hat{h}$ converges to $\mathcal{P}h$ faster for higher average spike rate (spike density $D$) of the neuron. At the same time, by increasing the stimulus bandwidth $\Omega$, we can approximate $h$ itself with arbitrary precision (Fig. 6(b)).

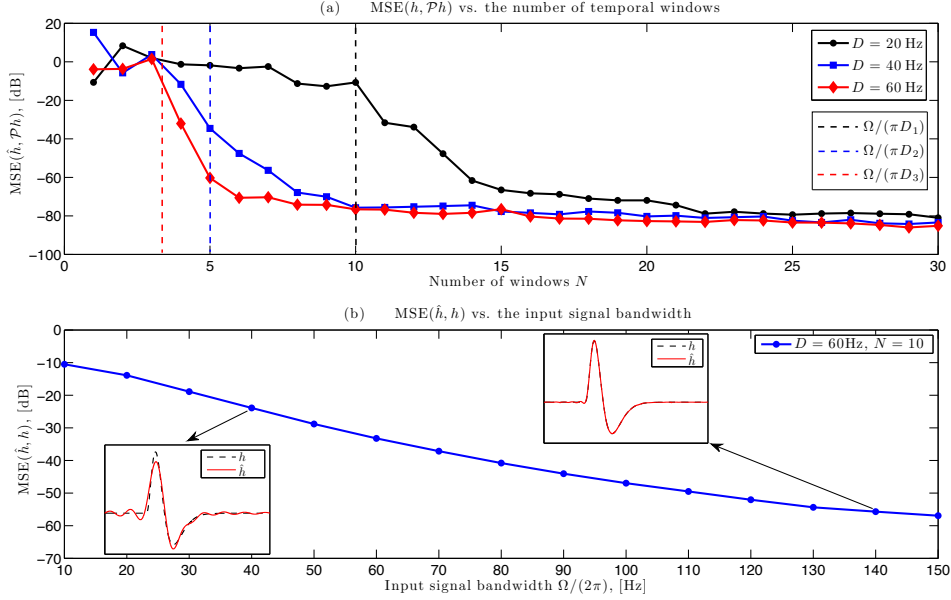

Figure 6: **The Filter Identification Error. (a)** $\text{MSE}(\hat{h}, \mathcal{P}h)$ as a function of the number of temporal windows $N$. The larger the neuron spike density $D$, the faster the algorithm converges. The impulse response $h$ is the same as in Fig. 4, 5 and the input signal bandwidth is $\Omega = 2\pi \cdot 100$ rad/s. **(b)** $\text{MSE}(\hat{h}, h)$ as a function of the input signal bandwidth $\Omega$. The larger the bandwidth, the better the estimate $\hat{h}$ approximates $h$. Note that significant improvement is seen even for $\Omega > 2\pi \cdot 100$ rad/s, which is roughly the effective bandwidth of $h$.

## 5 Conclusion

Previous work in system identification of neural circuits (see [20] and references therein) calls for parameter identification using white noise input stimuli. The identification process for, e.g., the LNP model entails identification of the linear filter, followed by a 'best-of-fit' procedure to find the non-linearity. The performance of such an identification method has not been analytically characterized.

In our work, we presented the methodology for identifying dendritic processing in simple [Filter]-[Spiking Neuron] models from a *single input stimulus*. The discussed spiking neurons include the ideal IAF neuron, the leaky IAF neuron and the threshold-and-fire neuron. However, the methods presented in this paper are applicable to many other spiking neuron models as well.

The algorithm of the Neuron Identification Machine is based on the natural assumption that the dendritic processing filter has a finite temporal support. Therefore, its action on the input stimulus can be observed in non-overlapping temporal windows. The filter is recovered with arbitrary precision from an input/output pair of a neural circuit, where the input is a *single signal* assumed to be bandlimited. Remarkably, the algorithm converges for a very small number of spikes. This should be contrasted with the reverse correlation and spike-triggered average methods [20].

Finally, the work presented here will be extended to spiking neurons with random parameters.

## Acknowledgement

The work presented here was supported by NIH under the grant number R01DC008701-01.

# References

[1] Maria N. Geffen, Bede M. Broome, Gilles Laurent, and Markus Meister. Neural encoding of rapidly fluctuating odors. Neuron, 61(4):570–586, 2009.

[2] Sean J. Slee, Matthew H. Higgs, Adrienne L. Fairhall, and William J. Spain. Two-dimensional time coding in the auditory brainstem. The Journal of Neuroscience, 25(43):9978–9988, October 2005.

[3] Nicole C. Rust, Odelia Schwartz, J. Anthony Movshon, and Eero P. Simoncelli. Spatiotemporal elements of macaque V1 receptive fields. Neuron, Vol. 46:945–956, 2005.

[4] Daniel P. Dougherty, Geraldine A. Wright, and Alice C. Yew. Computational model of the cAMP-mediated sensory response and calcium-dependent adaptation in vertebrate olfactory receptor neurons. Proceedings of the National Academy of Sciences, 102(30):0415–10420, 2005.

[5] Yuqiao Gu, Philippe Lucas, and Jean-Pierre Rospars. Computational model of the insect pheromone transduction cascade. PLoS Computational Biology, 5(3), 2009.

[6] Zhuoyi Song, Daniel Coca, Stephen Billings, Marten Postma, Roger C. Hardie, and Mikko Juusola. Biophysical Modeling of a Drosophila Photoreceptor. In Lecture Notes In Computer Science., volume 5863 of Proceedings of the 16th International Conference on Neural Information Processing: Part I, pages 57 – 71. Springer-Verlag, 2009.

[7] E.J. Chichilnisky. A simple white noise analysis of neuronal light responses. Network: Computation in Neural Systems, 12:199–213, 2001.

[8] Jonathan W. Pillow and Eero P. Simoncelli. Dimensionality reduction in neural models: An information-theoretic generalization of spike-triggered average and covariance analysis. Journal of Vision, 6:414–428, 2006.

[9] J J Eggermont, A M H J Aersten, and P I M Johannesma. Quantitative characterization procedure for auditory neurons based on the spectra-temporal receptive field. Hearing Research, 10, 1983.

[10] Anmo J. Kim, Aurel A. Lazar, and Yevgeniy B. Slutskiy. System identification of Drosophila olfactory sensory neurons. Journal of Computational Neuroscience, 2010.

[11] Sydney Cash and Rafael Yuste. Linear summation of excitatory inputs by CA1 pyramidal neurons. Neuron, 22:383–394, 1999.

[12] Jonathan Pillow. Neural coding and the statistical modeling of neuronal responses. PhD thesis, New York University, May 2005.

[13] Aurel A. Lazar and Laszlo T. Tóth. Perfect recovery and sensitivity analysis of time encoded bandlimited signals. IEEE Transactions on Circuits and Systems-I: Regular Papers, 51(10):2060–2073, October 2004.

[14] Aurel A. Lazar and Eftychios A. Pnevmatikakis. Faithful representation of stimuli with a population of integrate-and-fire neurons. Neural Computation, 20(11):2715–2744, November 2008.

[15] Justin Keat, Pamela Reinagel, R. Clay Reid, and Markus Meister. Predicting every spike: A model for the responses of visual neurons. Neuron, 30:803–817, June 2001.

[16] Michael Reed and Barry Simon. Methods of Modern Mathematical Physics, Vol. 1, Functional Analysis. Academic Press, 1980.

[17] Alain Berlinet and Christine Thomas-Agnan. Reproducing Kernel Hilbert Spaces in Probability and Statistics. Kluwer Academic Publishers, 2004.

[18] Ole Christensen. An Introduction to Frames and Riesz Bases. Applied and Numerical Harmonic Analysis. Birkhäuser, 2003.

[19] Edward H. Adelson and James R. Bergen. Spatiotemporal energy models for the perception of motion. Journal of Optical Society of America, 2(2), February 1985.

[20] Michael C.-K. Wu, Stephen V. David, and Jack L. Gallant. Complete functional characterization of sensory neurons by system identification. Annual Reviews of Neuroscience, 29:477–505, 2006.

